# An Input Output HMM Architecture

**Yoshua Bengio**[*]
Dept. Informatique et Recherche
Opérationnelle
Université de Montréal, Qc H3C-3J7
bengioy@IRO.UMontreal.CA

**Paolo Frasconi**
Dipartimento di Sistemi e Informatica
Università di Firenze (Italy)
paolo@mcculloch.ing.unifi.it

## Abstract

We introduce a recurrent architecture having a modular structure and we formulate a training procedure based on the EM algorithm. The resulting model has similarities to hidden Markov models, but supports recurrent networks processing style and allows to exploit the supervised learning paradigm while using maximum likelihood estimation.

## 1 INTRODUCTION

Learning problems involving sequentially structured data cannot be effectively dealt with static models such as feedforward networks. Recurrent networks allow to model complex dynamical systems and can store and retrieve contextual information in a flexible way. Up until the present time, research efforts of supervised learning for recurrent networks have almost exclusively focused on error minimization by gradient descent methods. Although effective for learning short term memories, practical difficulties have been reported in training recurrent neural networks to perform tasks in which the temporal contingencies present in the input/output sequences span long intervals (Bengio et al., 1994; Mozer, 1992).

Previous work on alternative training algorithms (Bengio et al., 1994) could suggest that the root of the problem lies in the essentially *discrete* nature of the process of storing information for an indefinite amount of time. Thus, a potential solution is to propagate, backward in time, targets in a discrete state space rather than differential error information. Extending previous work (Bengio & Frasconi, 1994a), in this paper we propose a statistical approach to target propagation, based on the EM algorithm. We consider a parametric dynamical system with discrete states and we introduce a modular architecture, with subnetworks associated to discrete states. The architecture can be interpreted as a statistical model and can be trained by the EM or generalized EM (GEM) algorithms (Dempster et al., 1977), considering the internal state trajectories as missing data. In this way learning is decoupled into

---

[*]also, AT&T Bell Labs, Holmdel, NJ 07733

a temporal credit assignment subproblem and a static learning subproblem that consists of fitting parameters to the next-state and output mappings defined by the estimated trajectories. In order to iteratively tune parameters with the EM or GEM algorithms, the system propagates forward and backward a discrete distribution over the $n$ states, resulting in a procedure similar to the Baum-Welch algorithm used to train standard hidden Markov models (HMMs) (Levinson et al., 1983). HMMs however adjust their parameters using unsupervised learning, whereas we use EM in a supervised fashion. Furthermore, the model presented here could be called *Input/Output HMM*, or IOHMM, because it can be used to learn to map input sequences to output sequences (unlike standard HMMs, which learn the output sequence distribution). This model can also be seen as a recurrent version of the Mixture of Experts architecture (Jacobs et al., 1991), related to the model already proposed in (Cacciatore and Nowlan, 1994). Experiments on artificial tasks (Bengio & Frasconi, 1994a) have shown that EM recurrent learning can deal with long term dependencies more effectively than backpropagation through time and other alternative algorithms. However, the model used in (Bengio & Frasconi, 1994a) has very limited representational capabilities and can only map an input sequence to a final discrete state. In the present paper we describe an extended architecture that allows to fully exploit both input and output portions of the data, as required by the supervised learning paradigm. In this way, general sequence processing tasks, such as production, classification, or prediction, can be dealt with.

## 2   THE PROPOSED ARCHITECTURE

We consider a discrete state dynamical system based on the following state space description:

$$
\begin{aligned}
x_t &= f(x_{t-1}, u_t) \\
y_t &= g(x_t, u_t)
\end{aligned}
\tag{1}
$$

where $u_t \in R^m$ is the input vector at time $t$, $y_t \in R^r$ is the output vector, and $x_t \in \{1, 2, \ldots, n\}$ is a discrete state. These equations define a generalized Mealy finite state machine, in which inputs and outputs may take on continuous values. In this paper, we consider a *probabilistic* version of these dynamics, where the current inputs and the current state distribution are used to estimate the state distribution and the output distribution for the next time step. Admissible state transitions will be specified by a directed graph $\mathcal{G}$ whose vertices correspond to the model's states and the set of successors for state $j$ is $\mathcal{S}_j$.

The system defined by equations (1) can be modeled by the recurrent architecture depicted in Figure 1(a). The architecture is composed by a set of *state networks* $\mathcal{N}_j, j = 1 \ldots n$ and a set of *output networks* $\mathcal{O}_j, j = 1 \ldots n$. Each one of the state and output networks is uniquely associated to one of the states, and all networks share the same input $u_t$. Each state network $\mathcal{N}_j$ has the task of predicting the next state distribution, based on the current input and given that $x_{t-1} = j$. Similarly, each output network $\mathcal{O}_j$ predicts the output of the system, given the current state and input. All the subnetworks are assumed to be static and they are defined by means of smooth mappings $N_j(u_t; \theta_j)$ and $O_j(u_t; \vartheta_j)$, where $\theta_j$ and $\vartheta_j$ are vectors of adjustable parameters (e.g., connection weights). The ranges of the functions $N_j()$ may be constrained in order to account for the underlying transition graph $\mathcal{G}$. Each output $\varphi_{ij,t}$ of the state subnetwork $\mathcal{N}_j$ (at time $t$) is associated to one of the successors $i$ of state $j$. Thus the last layer of $\mathcal{N}_j$ has as many units as the cardinality of $\mathcal{S}_j$. For convenience of notation, we suppose that $\varphi_{ij,t}$ are defined for each $i, j = 1, \ldots, n$ and we impose the condition $\varphi_{ij,t} = 0$ for each $i$ not belonging to $\mathcal{S}_j$. The *softmax* function is used in the last layer: $\varphi_{ij,t} = e^{a_{ij,t}} / \sum_{\ell \in \mathcal{S}_j} e^{a_{\ell j,t}}$, $j = 1, \ldots, n$, $i \in \mathcal{S}_j$ where $a_{ij,t}$ are intermediate variables that can be thought of as the

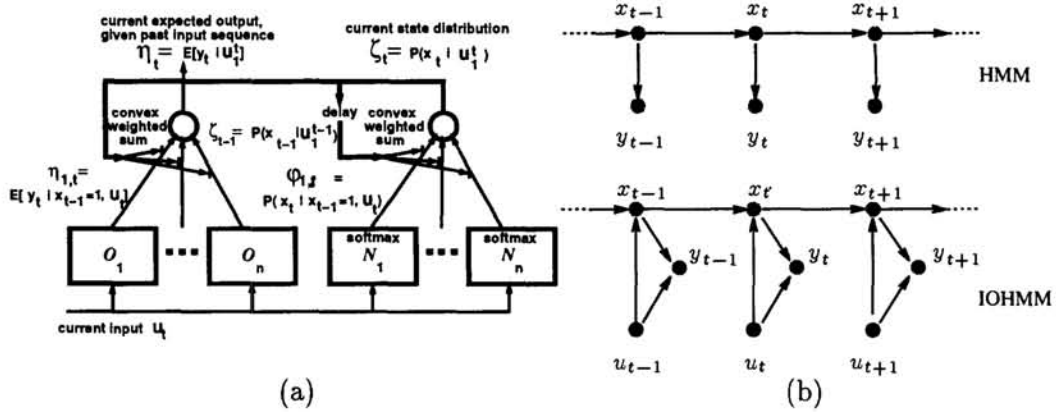

Figure 1: (a): The proposed IOHMM architecture. (b): Bottom: Bayesian network expressing conditional dependencies for an IOHMM; top: Bayesian network for a standard HMM

activations of the output units of subnetwork $\mathcal{N}_j$. In this way $\sum_{i=1}^{n} \varphi_{ij,t} = 1 \ \forall j, t$. The vector $\boldsymbol{\zeta}_t \in \boldsymbol{R}^n$ represents the internal state of the model and it is computed as a linear combination of the outputs of the state networks, gated by the previously computed internal state:

$$\boldsymbol{\zeta}_t = \sum_{j=1}^{n} \zeta_{j,t-1} \boldsymbol{\varphi}_{j,t} \tag{2}$$

where $\boldsymbol{\varphi}_{j,t} = [\varphi_{1j,t}, \ldots, \varphi_{nj,t}]'$. Output networks compete to predict the global output of the system $\boldsymbol{\eta}_t \in \boldsymbol{R}^r$:

$$\boldsymbol{\eta}_t = \sum_{j=1}^{n} \zeta_{jt} \boldsymbol{\eta}_{jt} \tag{3}$$

where $\boldsymbol{\eta}_{jt} \in \boldsymbol{R}^r$ is the output of subnetwork $\mathcal{O}_j$. At this level, we do not need to further specify the internal architecture of the state and output subnetworks. Depending on the task, the designer may decide whether to include hidden layers and what activation rule to use for the hidden units.

This connectionist architecture can be also interpreted as a probability model. Let us assume a multinomial distribution for the state variable $x_t$ and let us consider $\boldsymbol{\zeta}_t$, the main variable of the temporal recurrence (2). If we initialize the vector $\boldsymbol{\zeta}_0$ to positive numbers summing to 1, it can be interpreted as a vector of initial state probabilities. In general, we obtain relation $\zeta_{it} = \mathrm{P}(x_t = i \mid \boldsymbol{u}_1^t)$, having denoted with $\boldsymbol{u}_1^t$ the subsequence of inputs from time 1 to $t$, inclusively. Equation (2) then has the following probabilistic interpretation:

$$P(x_t = i \mid \boldsymbol{u}_1^t) = \sum_{j=1}^{n} P(x_t = i \mid x_{t-1} = j, \boldsymbol{u}_t) P(x_{t-1} = j \mid \boldsymbol{u}_1^{t-1}) \tag{4}$$

i.e., the subnetworks $\mathcal{N}_j$ compute transition probabilities conditioned on the input sequence $\boldsymbol{u}_t$:

$$\varphi_{ij,t} = \mathrm{P}(x_t = i \mid x_{t-1} = j, \boldsymbol{u}_t) \tag{5}$$

As in neural networks trained to minimize the output squared error, the output $\boldsymbol{\eta}_t$ of this architecture can be interpreted as an expected "position parameter" for the probability distribution of the output $\boldsymbol{y}_t$. However, in addition to being conditional on an input $\boldsymbol{u}_t$, this expectation is also conditional on the state $x_t$, i.e.

$\eta_t = E[\boldsymbol{y}_t \mid x_t, \boldsymbol{u}_t]$. The actual form of the output density, denoted $f_Y(\boldsymbol{y}_t; \eta_t)$, will be chosen according to the task. For example a multinomial distribution is suitable for sequence classification, or for symbolic mutually exclusive outputs. Instead, a Gaussian distribution is adequate for producing continuous outputs. In the first case we use a softmax function at the output of subnetworks $\mathcal{O}_j$; in the second case we use linear output units for the subnetworks $\mathcal{O}_j$.

In order to reduce the amount of computation, we introduce an independency model among the variables involved in the probabilistic interpretation of the architecture. We shall use a Bayesian network to characterize the probabilistic dependencies among these variables. Specifically, we suppose that the directed acyclic graph $\mathcal{G}$ depicted at the bottom of Figure 1b is a Bayesian network for the dependency model associated to the variables $\boldsymbol{u}_1^T, x_1^T, \boldsymbol{y}_1^T$. One of the most evident consequences of this independency model is that only the previous state and the current input are relevant to determine the next-state. This one-step memory property is analogue to the Markov assumption in hidden Markov models (HMM). In fact, the Bayesian network for HMMs can be obtained by simply removing the $\boldsymbol{u}_t$ nodes and arcs from them (see top of Figure 1b).

## 3  A SUPERVISED LEARNING ALGORITHM

The learning algorithm for the proposed architecture is derived from the maximum likelihood principle. The training data are a set of $P$ pairs of input/output sequences (of length $T_p$): $\mathcal{D} = \{(\boldsymbol{u}_1^{T_p}(p), \boldsymbol{y}_1^{T_p}(p)); p = 1 \ldots P\}$. Let $\boldsymbol{\Theta}$ denote the vector of parameters obtained by collecting all the parameters $\boldsymbol{\theta}_j$ and $\boldsymbol{\vartheta}_i$ of the architecture. The likelihood function is then given by

$$L(\boldsymbol{\Theta}; \mathcal{D}) = \prod_{p=1}^{P} \mathrm{P}(\boldsymbol{y}_1^{T_p}(p) \mid \boldsymbol{u}_1^{T_p}(p); \boldsymbol{\Theta}). \tag{6}$$

The output values (used here as targets) may also be specified intermittently. For example, in sequence classification tasks, one may only be interested in the output $\boldsymbol{y}_T$ at the end of each sequence. The modification of the likelihood to account for intermittent targets is straightforward. According to the maximum likelihood principle, the optimal parameters are obtained by maximizing (6). In order to apply EM to our case we begin by noting that the state variables $x_t$ are not observed. Knowledge of the model's state trajectories would allow one to decompose the temporal learning problem into $2n$ *static* learning subproblems. Indeed, if $x_t$ were known, the probabilities $\zeta_{it}$ would be either 0 or 1 and it would be possible to train each subnetwork separately, without taking into account any temporal dependency. This observation allows to link EM learning to the target propagation approach discussed in the introduction. Note that if we used a Viterbi-like approximation (i.e., considering only the most likely path), we would indeed have $2n$ static learning problems at each epoch. In order to we derive the learning equations, let us define the *complete data* as $\mathcal{D}_c = \{(\boldsymbol{u}_1^{T_p}(p), \boldsymbol{y}_1^{T_p}(p), x_1^{T_p}(p)); p = 1 \ldots P\}$. The corresponding complete data log-likelihood is

$$l_c(\boldsymbol{\Theta}; \mathcal{D}_c) = \sum_{p=1}^{P} \log \mathrm{P}(\boldsymbol{y}_1^{T_p}(p), \boldsymbol{z}_1^{T_p}(p) \mid \boldsymbol{u}_1^{T_p}(p); \boldsymbol{\Theta}). \tag{7}$$

Since $l_c(\boldsymbol{\Theta}; \mathcal{D}_c)$ depends on the hidden state variables it cannot be maximized directly. The MLE optimization is then solved by introducing the auxiliary function $Q(\boldsymbol{\Theta}; \hat{\boldsymbol{\Theta}})$ and iterating the following two steps for $k = 1, 2, \ldots$:

Estimation:    Compute $Q(\boldsymbol{\Theta}; \hat{\boldsymbol{\Theta}}) = E[l_c(\boldsymbol{\Theta}; \mathcal{D}_c) \mid \mathcal{D}, \hat{\boldsymbol{\Theta}}]$

Maximization:  Update the parameters as $\hat{\boldsymbol{\Theta}} \leftarrow \arg\max_{\boldsymbol{\Theta}} Q(\boldsymbol{\Theta}; \hat{\boldsymbol{\Theta}})$

$$\tag{8}$$

The expectation of (7) can be expressed as

$$Q(\boldsymbol{\Theta}; \hat{\boldsymbol{\Theta}}) = \sum_{p=1}^{P} \sum_{t=1}^{T_p} \sum_{i=1}^{N} \hat{\zeta}_{it} \log \mathrm{P}(\boldsymbol{y}_t \mid x_t = i, \boldsymbol{u}_t; \boldsymbol{\Theta}) + \sum_{j=1}^{N} \hat{h}_{ij,t} \log \varphi_{ij,t} \qquad (9)$$

where $h_{ij,t} = E[z_{it} z_{j,t-1} \mid \boldsymbol{u}_1^T, \boldsymbol{y}_1^T; \boldsymbol{\Theta}]$, denoting $z_{it}$ for an indicator variable $= 1$ if $x_t = i$ and 0 otherwise. The hat in $\hat{\zeta}_{it}$ and $\hat{h}_{ij,t}$ means that these variables are computed using the "old" parameters $\hat{\boldsymbol{\Theta}}$. In order to compute $h_{ij,t}$ we introduce the forward probabilities $\alpha_{it} = \mathrm{P}(\boldsymbol{y}_1^t, x_t = i; \boldsymbol{u}_1^t)$ and the backward probabilities $\beta_{it} = \mathrm{P}(\boldsymbol{y}_t^T \mid x_t = i, \boldsymbol{u}_t^T)$, that are updated as follows:

$$\beta_{it} = f_Y(\boldsymbol{y}_t; \boldsymbol{\eta}_{it}) \sum_\ell \varphi_{\ell i}(\boldsymbol{u}_{t+1}) \beta_{\ell,t+1}$$

$$\alpha_{it} = f_Y(\boldsymbol{y}_t; \boldsymbol{\eta}_{it}) \sum_\ell \varphi_{i\ell}(\boldsymbol{u}_t) \alpha_{\ell,t-1}. \qquad (10)$$

$$h_{ij,t} = \frac{\beta_{it} \alpha_{j,t-1} \varphi_{ij}(\boldsymbol{u}_t)}{\sum_i \alpha_{iT}} \qquad (11)$$

Each iteration of the EM algorithm requires to maximize $Q(\boldsymbol{\Theta}; \hat{\boldsymbol{\Theta}})$. We first consider a simplified case, in which the inputs are quantized (i.e., belonging to a finite alphabet $\{\sigma_1, \ldots, \sigma_K\}$) and the subnetworks behave like lookup tables addressed by the input symbols $\sigma_t$, i.e. we interpret each parameter as $w_{ijk} = \mathrm{P}(x_t = i \mid x_{t-1} = j, \sigma_t = k)$. For simplicity, we restrict the analysis to classification tasks and we suppose that targets are specified as desired final states for each sequence. Furthermore, no output subnetworks are used in this particular application of the algorithm. In this case we obtain the reestimation formulae:

$$w_{ijk} = \frac{\sum_{p=1}^{P} \sum_{t:\sigma_t=k} \frac{\hat{\beta}_{it} \hat{\zeta}_{j,t-1}}{\hat{\zeta}_{x_T^\star, T}}}{\sum_{i \in \mathcal{S}_j} \sum_{p=1}^{P} \sum_{t:\sigma_t=k} \frac{\hat{\beta}_{it} \hat{\zeta}_{j,t-1}}{\hat{\zeta}_{x_T^\star, T}}}. \qquad (12)$$

In general, however, if the subnetworks have hidden sigmoidal units, or use a softmax function to constrain their outputs to sum to one, the maximum of $Q$ cannot be found analytically. In these cases we can resort to a GEM algorithm, that simply produces an increase in $Q$, for example by gradient ascent. In this case, the derivatives of $Q$ with respect to the parameters can be easily computed as follows. Let $\theta_{jk}$ be a generic weight in the state subnetwork $\mathcal{N}_j$. From equation (9):

$$\frac{\partial Q(\boldsymbol{\Theta}; \hat{\boldsymbol{\Theta}})}{\partial \theta_{jk}} = \sum_p \sum_t \sum_i \hat{h}_{ij,t} \frac{1}{\varphi_{ij,t}} \frac{\partial \varphi_{ij,t}}{\partial \theta_{jk}} \qquad (13)$$

where the partial derivatives $\frac{\partial \varphi_{ij,t}}{\partial \theta_{jk}}$ can be computed using backpropagation. Similarly, denoting with $\vartheta_{ik}$ a generic weight of the output subnetwork $\mathcal{O}_i$, we have:

$$\frac{\partial Q(\boldsymbol{\Theta}; \hat{\boldsymbol{\Theta}})}{\partial \vartheta_{ik}} = \sum_p \sum_t \sum_\ell \hat{\zeta}_{i,t} \frac{\partial}{\partial \eta_{i\ell,t}} \log f_Y(\boldsymbol{y}_y; \boldsymbol{\eta}_{it}) \frac{\partial \eta_{i\ell,t}}{\partial \vartheta_{ik}} \qquad (14)$$

where $\frac{\partial \eta_{i\ell,t}}{\partial \vartheta_{ik}}$ are also computed using backpropagation. Intuitively, the parameters are updated as if the estimation step of EM had provided targets for the outputs of the $2n$ subnetworks, for each time $t$. Although GEM algorithms are also guaranteed to find a local maximum of the likelihood, their convergence may be significantly slower compared to EM. In several experiments we noticed that convergence can be accelerated with stochastic gradient ascent.

## 4   COMPARISONS

It appears natural to find similarities between the recurrent architecture described so far and standard HMMs (Levinson et al., 1983). The architecture proposed in this paper differs from standard HMMs in two respects: computing style and learning. With IOHMMs, sequences are processed similarly to recurrent networks, e.g., an input sequence can be synchronously transformed into an output sequence. This computing style is real-time and predictions of the outputs are available as the input sequence is being processed. This architecture thus allows one to implement all three fundamental sequence processing tasks: *production*, *prediction*, and *classification*. Finally, transition probabilities in standard HMMs are fixed, i.e. states form a *homogeneous* Markov chain. In IOHMMs, transition probabilities are conditional on the input and thus depend on time, resulting in an *inhomogeneous* Markov chain. Consequently, the *dynamics* of the system (specified by the transition probabilities) are not fixed but are *adapted* in time depending on the input sequence.

The other fundamental difference is in the learning procedure. While interesting for their capabilities of modeling sequential phenomena, a major weakness of standard HMMs is their poor discrimination power due to unsupervised learning. An approach that has been found useful to improve discrimination in HMMs is based on maximum mutual information (MMI) training. It has been pointed out that supervised learning and discriminant learning criteria like MMI are actually strictly related (Bridle, 1989). Although the parameter adjusting procedure we have defined is based on MLE, $y_1^T$ is used as *desired output* in response to the input $u_1^T$, resulting in discriminant supervised learning. Finally, it is worth mentioning that a number of hybrid approaches have been proposed to integrate connectionist approaches into the HMM framework. For example in (Bengio et al., 1992) the observations used by the HMM are generated by a feedforward neural network. In (Bourlard and Wellekens, 1990) a feedforward network is used to estimate state probabilities, conditional to the acoustic sequence. A common feature of these algorithms and the one proposed in this paper is that neural networks are used to extract temporally local information whereas a Markovian system integrates long-term constraints.

We can also establish a link between IOHMMs and adaptive mixtures of experts (ME) (Jacobs et al., 1991). Recently, Cacciatore & Nowlan (1994) have proposed a recurrent extension to the ME architecture, called *mixture of controllers* (MC), in which the gating network has feedback connections, thus allowing to take temporal context into account. Our IOHMM architecture can be interpreted as a special case of the MC architecture, in which the set of state subnetworks play the role of a gating network having a modular structure and second order connections.

## 5   REGULAR GRAMMAR INFERENCE

In this section we describe an application of our architecture to the problem of grammatical inference. In this task the learner is presented a set of labeled strings and is requested to infer a set of rules that define a formal language. It can be considered as a prototype for more complex language processing problems. However, even in the "simplest" case, i.e. regular grammars, the task can be proved to be NP-complete (Angluin and Smith, 1983). We report experimental results on a set of regular grammars introduced by Tomita (1982) and afterwards used by other researchers to measure the accuracy of inference methods based on recurrent networks (Giles et al., 1992; Pollack, 1991; Watrous and Kuhn, 1992).

We used a scalar output with supervision on the final output $y_T$ that was modeled as a Bernoulli variable $f_Y(y_T; \eta_T) = \eta_T^{y_T}(1 - \eta_T)^{1-y_T}$, with $y_T = 0$ if the string is rejected and $y_T = 1$ if it is accepted. In this application we did not apply

Table 1: Summary of experimental results on the seven Tomita's grammars.

| Grammar | Sizes | | Convergence | Accuracies | | | |
|---|---|---|---|---|---|---|---|
| | $n^\star$ | FSA min | | Average | Worst | Best | W&K Best |
| 1 | 2 | 2 | .600 | 1.000 | 1.000 | 1.000 | 1.000 |
| 2 | 8 | 3 | .800 | .965 | .834 | 1.000 | 1.000 |
| 3 | 7 | 5 | .150 | .867 | .775 | 1.000 | .783 |
| 4 | 4 | 4 | .100 | 1.000 | 1.000 | 1.000 | .609 |
| 5 | 4 | 4 | .100 | 1.000 | 1.000 | 1.000 | .668 |
| 6 | 3 | 3 | .350 | 1.000 | 1.000 | 1.000 | .462 |
| 7 | 3 | 5 | .450 | .856 | .815 | 1.000 | .557 |

external inputs to the output networks. This corresponds to modeling a Moore finite state machine. Given the absence of prior knowledge about plausible state paths, we used an *ergodic* transition graph (i.e., fully connected).In the experiments we measured convergence and generalization performance using different sizes for the recurrent architecture. For each setting we ran 20 trials with different seeds for the initial weights. We considered a trial successful if the trained network was able to correctly label all the training strings. The model size was chosen using a cross-validation criterion based on performance on 20 randomly generated strings of length $T \leq 12$. For comparison, in Table 1 we also report for each grammar the number of states of the minimal recognizing FSA (Tomita, 1982). We tested the trained networks on a corpus of $2^{13} - 1$ binary strings of length $T \leq 12$. The final results are summarized in Table 1. The column "Convergence" reports the fraction of trials that succeeded to separate the training set. The next three columns report averages and order statistics (worst and best trial) of the fraction of correctly classified strings, measured on the successful trials. For each grammar these results refer to the model size $n^\star$ selected by cross-validation. Generalization was always perfect on grammars 1,4,5 and 6. For each grammar, the best trial also attained perfect generalization. These results compare very favorably to those obtained with second-order networks trained by gradient descent, when using the learning sets proposed by Tomita. For comparison, in the last column of Table 1 we reproduce the results reported by Watrous & Kuhn (1992) in the best of five trials. In most of the successful trials the model learned an actual FSA behavior with transition probabilities asymptotically converging either to 0 or to 1. This renders trivial the extraction of the corresponding FSA. Indeed, for grammars 1,4,5, and 6, we found that the trained networks behave exactly like the minimal recognizing FSA.

A potential training problem is the presence of local maxima in the likelihood function. For example, the number of converged trials for grammars 3, 4, and 5 is quite small and the difficulty of discovering the optimal solution might become a serious restriction for tasks involving a large number of states. In other experiments (Bengio & Frasconi, 1994a), we noticed that restricting the connectivity of the transition graph can significantly help to remove problems of convergence. Of course, this approach can be effectively exploited only if some prior knowledge about the state space is available. For example, applications of HMMs to speech recognition always rely on structured topologies.

# 6  CONCLUSIONS

There are still a number of open questions. In particular, the effectiveness of the model on tasks involving large or very large state spaces needs to be carefully evaluated. In (Bengio & Frasconi 1994b) we show that learning long term dependencies in these models becomes more difficult as we increase the connectivity of the state

transition graph. However, because transition probabilities of IOHMMs change at each $t$, they deal better with this problem of long-term dependencies than standard HMMs. Another interesting aspect to be investigated is the capability of the model to successfully perform tasks of sequence production or prediction. For example, interesting tasks that could also be approached are those related to time series modeling and motor control learning.

## References

Angluin, D. and Smith, C. (1983). Inductive inference: Theory and methods. *Computing Surveys*, 15(3):237–269.

Bengio, Y. and Frasconi, P. (1994a). Credit assignment through time: Alternatives to backpropagation. In Cowan, J., Tesauro, G., and Alspector, J., editors, *Advances in Neural Information Processing Systems 6*. Morgan Kaufmann.

Bengio, Y. and Frasconi, P. (1994b). An EM Approach to Learning Sequential Behavior. Tech. Rep. RT-DSI/11-94, University of Florence.

Bengio, Y., De Mori, R., Flammia, G., and Kompe, R. (1992). Global optimization of a neural network-hidden markov model hybrid. *IEEE Transactions on Neural Networks*, 3(2):252–259.

Bengio, Y., Simard, P., and Frasconi, P. (1994). Learning long-term dependencies with gradient descent is difficult. *IEEE Trans. Neural Networks*, 5(2).

Bourlard, H. and Wellekens, C. (1990). Links between hidden markov models and multilayer perceptrons. *IEEE Trans. Pattern An. Mach. Intell.*, 12:1167–1178.

Bridle, J. S. (1989). Training stochastic model recognition algorithms as networks can lead to maximum mutual information estimation of parameters. In D.S.Touretzky, ed., *NIPS2*, pages 211–217. Morgan Kaufmann.

Cacciatore, T. W. and Nowlan, S. J. (1994). Mixtures of controllers for jump linear and non-linear plants. In Cowan, J. et. al., editors, *Advances in Neural Information Processing Systems 6*, San Mateo, CA. Morgan Kaufmann.

Dempster, A. P., Laird, N. M., and Rubin, D. B. (1977). Maximum-likelihood from incomplete data via the EM algorithm. *J. Royal Stat. Soc. B*, 39:1–38.

Giles, C. L., Miller, C. B., Chen, D., Sun, G. Z., Chen, H. H., and Lee, Y. C. (1992). Learning and extracting finite state automata with second-order recurrent neural networks. *Neural Computation*, 4(3):393–405.

Jacobs, R. A., Jordan, M. I., Nowlan, S. J., and Hinton, G. E. (1991). Adaptive mixture of local experts. *Neural Computation*, 3:79–87.

Levinson, S. E., Rabiner, L. R., and Sondhi, M. M. (1983). An introduction to the application of the theory of probabilistic functions of a markov process to automatic speech recognition. *Bell System Technical Journal*, 64(4):1035–1074.

Mozer, M. C. (1992). The induction of multiscale temporal structure. In Moody, J. et. al., eds, *NIPS* 4 pages 275–282. Morgan Kaufmann.

Pollack, J. B. (1991). The induction of dynamical recognizers. *Machine Learning*, 7(2):196–227.

Tomita, M. (1982). Dynamic construction of finite-state automata from examples using hill-climbing. *Proc. 4th Cog. Science Conf.*, pp. 105–108, Ann Arbor MI.

Watrous, R. L. and Kuhn, G. M. (1992). Induction of finite-state languages using second-order recurrent networks. *Neural Computation*, 4(3):406–414.
